# Unsupervised learning
# of distributions on binary vectors
# using two layer networks

**Yoav Freund** *
Computer and Information Sciences
University of California Santa Cruz
Santa Cruz, CA 95064

**David Haussler**
Computer and Information Sciences
University of California Santa Cruz
Santa Cruz, CA 95064

## Abstract

We study a particular type of Boltzmann machine with a bipartite graph structure called a harmonium. Our interest is in using such a machine to model a probability distribution on binary input vectors. We analyze the class of probability distributions that can be modeled by such machines, showing that for each $n \geq 1$ this class includes arbitrarily good approximations to any distribution on the set of all $n$-vectors of binary inputs. We then present two learning algorithms for these machines. The first learning algorithm is the standard gradient ascent heuristic for computing maximum likelihood estimates for the parameters (i.e. weights and thresholds) of the model. Here we give a closed form for this gradient that is significantly easier to compute than the corresponding gradient for the general Boltzmann machine. The second learning algorithm is a greedy method that creates the hidden units and computes their weights one at a time. This method is a variant of the standard method for projection pursuit density estimation. We give experimental results for these learning methods on synthetic data and natural data from the domain of handwritten digits.

## 1  Introduction

Let us suppose that each example in our input data is a binary vector $\vec{x} = \{x_1, \dots, x_n\} \in \{\pm 1\}^n$, and that each such example is generated independently at random according some unknown distribution on $\{\pm 1\}^n$. This situation arises, for instance, when each example consists of (possibly noisy) measurements of $n$ different binary attributes of a randomly selected object. In such a situation, unsupervised learning can be usefully defined as using the input data to find a good model of the unknown distribution on $\{\pm 1\}^n$ and thereby learning the structure in the data.

The process of learning an unknown distribution from examples is usually called *density estimation* or *parameter estimation* in statistics, depending on the nature of the class of distributions used as models. Connectionist models of this type include Bayes networks [14], mixture models [3,13], and Markov random fields [14,8]. Network models based on the notion of energy minimization such as Hopfield nets [9] and Boltzmann machines [1] can also be used as models of probability distributions.

The models defined by Hopfield networks are a special case of the more general Markov random field models in which the local interactions are restricted to symmetric pairwise interactions between components of the input. Boltzmann machines also use only pairwise interactions, but in addition they include *hidden units*, which correspond to unobserved variables. These unobserved variables interact with the observed variables represented by components of the input vector. The overall distribution on the set of possible input vectors is defined as the marginal distribution induced on the components of the input vector by the Markov random field over all variables, both observed and hidden. While the Hopfield network is relatively well understood, it is limited in the types of distributions that it can model. On the other hand, Boltzmann machines are universal in the sense that they are powerful enough to model any distribution (to any degree of approximation), but the mathematical analysis of their capabilities is often intractable. Moreover, the standard learning algorithm for the Boltzmann machine, a gradient ascent heuristic to compute the maximum likelihood estimates for the weights and thresholds, requires repeated stochastic approximation, which results in unacceptably slow learning. [1] In this work we attempt to narrow the gap between Hopfield networks and Boltzmann machines by finding a model that will be powerful enough to be universal, [2] yet simple enough to be analyzable and computationally efficient. [3] We have found such a model in a minor variant of the special type of Boltzmann machine defined by Smolensky in his *harmony theory* [16][Ch.6]. This special type of Boltzmann machine is defined by a network with a simple bipartite graph structure, which he called a *harmonium*.

The harmonium consists of two types of units: input units, each of which holds one component of the input vector, and hidden units, representing hidden variables. There is a weighted connection between each input unit and each hidden unit, and no connections between input units or between hidden units (see Figure (1)). The presence of the hidden units induces dependencies, or correlations, between the variables modeled by input units. To illustrate the kind of model that results, consider the distribution of people that visit a specific coffee shop on Sunday. Let each of the $n$ input variables represent the presence ($+1$) or absence ($-1$) of a particular person that Sunday. These random variables are clearly not independent, e.g. if Fred's wife and daughter are there, it is more likely that Fred is there, if you see three members of the golf club, you expect to see other members of the golf club, if Bill is there you are unlikely to see Brenda there, etc. This situation can be modeled by a harmonium model in which each hidden variable represents the presence or absence of a social group. The weights connecting a hidden unit and an input unit measure the tendency of the corresponding person to be associated with the corresponding group. In this coffee shop situation, several social groups may be present at one time, exerting a combined influence on the distribution of customers. This can be modeled easily with the harmonium, but is difficult to model using Bayes networks or mixture models. [4]

## 2    The Model

Let us begin by formalizing the harmonium model. To model a distribution on $\{\pm1\}^n$ we will use $n$ input units and some number $m \geq 0$ of hidden units. These units are connected in a bipartite graph as illustrated in Figure (1).

The random variables represented by the input units each take values in $\{+1, -1\}$, while the hidden variables, represented by the hidden units, take values in $\{0, 1\}$. The state of the machine is defined by the values of these random variables. Define $\vec{x} = (x_1, \ldots, x_n) \in \{\pm1\}^n$ to be the state of the input units, and $\vec{h} = (h_1, \ldots, h_m) \in \{0, 1\}^m$ to be the state of the hidden units.

The connection weights between the input units and the $i$th hidden unit are denoted [5] by $\vec{\omega}^{(i)} \in R^n$ and the bias of the $i$th hidden unit is denoted by $\theta^{(i)} \in R$. The parameter vector $\phi = \{(\vec{\omega}^{(1)}, \theta^{(1)}), \ldots, (\vec{\omega}^{(m)}, \theta^{(m)})\}$

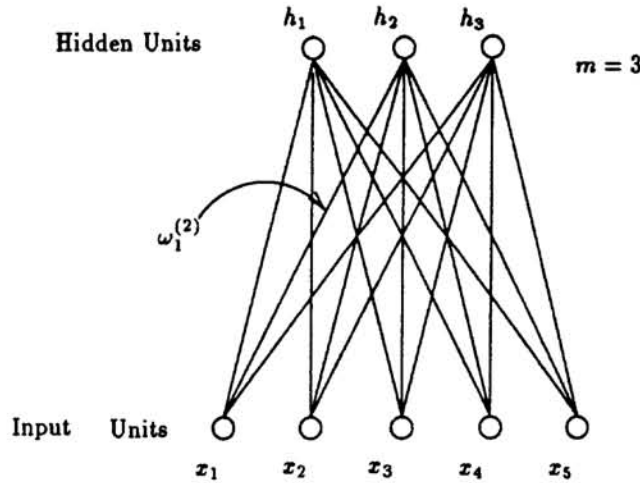

Figure 1: The bipartite graph of the harmonium

defines the entire network, and thus also the probability model induced by the network. For a given $\phi$, the energy of a state configuration of hidden and input units is defined to be

$$E(\vec{x}, \vec{h}|\phi) = -\sum_{i=1}^{m}(\vec{\omega}^{(i)} \cdot \vec{x} + \theta^{(i)})h_i \qquad (1)$$

and the probability of a configuration is

$$Pr(\vec{x}, \vec{h}|\phi) = \frac{1}{Z}e^{-E(\vec{x}, \vec{h}|\phi)} \quad \text{where} \quad Z = \sum_{\vec{x}, \vec{h}} e^{-E(\vec{x}, \vec{h}|\phi)}.$$

Summing over $\vec{h}$, it is easy to show that in the general case the probability distribution over possible state vectors on the input units is given by

$$Pr(\vec{x}|\phi) = \sum_{\vec{h}\in\{0,1\}^m} Pr(\vec{x}, \vec{h}|\phi) = \frac{1}{Z}\sum_{\vec{h}\in\{0,1\}^m} \exp\left(\sum_{i=1}^{m}(\vec{\omega}^{(i)} \cdot \vec{x} + \theta^{(i)})h_i\right) = \frac{1}{Z}\prod_{i=1}^{m}\left(1 + e^{\vec{\omega}^{(i)}\cdot\vec{x}+\theta^{(i)}}\right), \quad (2)$$

This product form is particular to the harmonium structure, and does not hold for general Boltzmann machines. Product form distribution models have been used for density estimation in Projection Pursuit [10,6,5]. We shall look further into this relationship in section 5.

## 3   Discussion of the model

The right hand side of Equation (2) has a simple intuitive interpretation. The $i$th factor in the product corresponds to the hidden variable $h_i$ and is an increasing function of the dot product between $\vec{x}$ and the weight vector of the $i$th hidden unit. Hence an input vector $\vec{x}$ will tend to have large probability when it is in the direction of one of the weight vectors $\vec{\omega}^{(i)}$ (i.e. when $\vec{\omega}^{(i)} \cdot \vec{x}$ is large), and small probability otherwise. This is the way that the hidden variables can be seen to exert their "influence"; each corresponds to a preferred or "prototypical" direction in space.

The next to the last formula in Equation (2) shows that the harmonium model can be written as a mixture of $2^m$ distributions of the form

$$\frac{1}{Z(\vec{h})}\exp\left(\sum_{i=1}^{m}(\vec{\omega}^{(i)} \cdot \vec{x} + \theta^{(i)})h_i\right),$$

where $\bar{h} \in \{0,1\}^m$ and $Z(\bar{h})$ is the appropriate normalization factor. It is easily verified that each of these distributions is in fact a product of $n$ Bernoulli distributions on $\{+1, -1\}$, one for each input variable $x_j$. Hence the harmonium model can be interpreted as a kind of mixture model. However, the number of components in the mixture represented by a harmonium is exponential in the number of hidden units.

It is interesting to compare the class of harmonium models to the standard class of models defined by a mixture of products of Bernoulli distributions. The same bipartite graph described in Figure (1) can be used to define a standard mixture model. Assign each of the $m$ hidden units a weight vector $\vec{w}^{(i)}$ and a probability $p_i$ such that $\sum_{i=1}^{m} p_i = 1$. To generate an example, choose *one* of the hidden units according to the distribution defined by the $p_i$'s, and then choose the vector $\vec{x}$ according to $P_i(\vec{x}) = \frac{1}{Z_i} e^{\vec{w}^{(i)} \cdot \vec{x}}$, where $Z_i$ is the appropriate normalization factor so that $\sum_{\vec{x} \in \{\pm 1\}^n} P_i(\vec{x}) = 1$. We thus get the distribution

$$P(\vec{x}) = \sum_{i=1}^{m} \frac{p_i}{Z_i} e^{\vec{w}^{(i)} \cdot \vec{x}} \qquad (3)$$

This form for presenting the standard mixture model emphasizes the similarity between this model and the harmonium model. A vector $\vec{x}$ will have large probability if the dot product $\vec{w}^{(i)} \cdot \vec{x}$ is large for some $1 \le i \le m$ (so long as $p_i$ is not too small). However, unlike the standard mixture model, the harmonium model allows more than one hidden variable to be $+1$ for any generated example. This means that several hidden influences can combine in the generation of a single example, because several hidden variables can be $+1$ at the same time. To see why this is useful, consider the coffee shop example given in the introduction. At any moment of time it is reasonable to find *several* social groups of people sitting in the shop. The harmonium model will have a natural representation for this situation, while in order for the standard mixture model to describe it accurately, a hidden variable has to be assigned to each combination of social groups that is likely to be found in the shop at the same time. In such cases the harmonium model is exponentially more succinct than the standard mixture model.

## 4    Learning by gradient ascent on the log-likelihood

We now suppose that we are given a sample consisting of a set $S$ of vectors in $\{\pm 1\}^n$ drawn independently at random from some unknown distribution. Our goal is use the sample $S$ to find a good model for this unknown distribution using a harmonium with $m$ hidden units, if possible. The method we investigate here is the method of maximum likelihood estimation using gradient ascent. The goal of learning is thus reduced to finding the set of parameters for the harmonium that maximize the (log of the) probability of the set of examples $S$. In fact, this gives the standard learning algorithm for general Boltzmann machines. For a general Boltzmann machine this would require stochastic estimation of the parameters. As stochastic estimation is very time-consuming, the result is that learning is very slow. In this section we show that stochastic estimation need not be used for the harmonium model.

From (2), the log likelihood of a sample of input vectors $S = \{\vec{x}^{(1)}, \vec{x}^{(2)}, \ldots, \vec{x}^{(N)}\}$, given a particular setting $\phi = \{(\vec{w}^{(1)}, \theta^{(1)}), \ldots, (\vec{w}^{(m)}, \theta^{(m)})\}$ of the parameters of the model is:

$$\text{log-likelihood}(\phi) = \sum_{\vec{x} \in S} \ln Pr(\vec{x}|\phi) = \sum_{i=1}^{m} \left( \sum_{\vec{x} \in S} \ln(1 + e^{\vec{w}^{(i)} \cdot \vec{x} + \theta^{(i)}}) \right) - N \ln Z . \qquad (4)$$

Taking the gradient of the log-likelihood results in the following formula for the $j$th component of $\vec{w}^{(i)}$

$$\frac{\partial}{\partial w_j^{(i)}} \text{log-likelihood}(\phi) = \sum_{\vec{x} \in S} x_j \frac{1}{1 + e^{-(\vec{w}^{(i)} \cdot \vec{x} + \theta^{(i)})}} - N \sum_{\vec{x} \in \{\pm 1\}^n} Pr(\vec{x}|\phi) x_j \frac{1}{1 + e^{-(\vec{w}^{(i)} \cdot \vec{x} + \theta^{(i)})}} \qquad (5)$$

A similar formula holds for the derivative of the bias term.

The purpose of the clamped and unclamped phases in the Boltzmann machine learning algorithm is to approximate these two terms. In general, this requires stochastic methods. However, here the clamped term is easy to calculate, it requires summing a logistic type function over all training examples. The same term

is obtained by making the mean field approximation for the clamped phase in the general algorithm [15], which is exact in this case. It is more difficult to compute the sleep phase term, as it is an explicit sum over the entire input space, and within each term of this sum there is an implicit sum over the entire space of configurations of hidden units in the factor $Pr(\vec{x}|\phi)$. However, again taking advantage of the special structure of the harmonium, we can reduce this sleep phase gradient term to a sum only over the configurations of the hidden units, yielding for each component of $\vec{\omega}^{(i)}$

$$\frac{\partial}{\partial \omega_j^{(i)}}\text{log-likelihood}(\phi) = \sum_{\vec{x} \in S} x_j \frac{1}{1 + e^{-(\vec{\omega}^{(i)} \cdot \vec{x} + \theta^{(i)})}} - N \sum_{\vec{h} \in \{0,1\}^m} Pr(\vec{h}|\phi) h_i \, \tanh(\sum_{k=1}^m h_k \omega_j^{(k)}) \qquad (6)$$

where

$$Pr(\vec{h}|\phi) = \frac{\exp(\sum_{i=1}^m h_i \theta^{(i)}) \prod_{j=1}^n \cosh(\sum_{i=1}^m h_i \omega_j^{(i)})}{\sum_{\vec{h}' \in \{0,1\}^m} \left[ \exp(\sum_{i=1}^m h_i' \theta^{(i)}) \prod_{j=1}^n \cosh(\sum_{i=1}^m h_i' \omega_j^{(i)}) \right]}$$

Direct computation of (6) is fast for small $m$ in contrast to the case for general Boltzmann machines (we have performed experiments with $m \le 10$). However, for large $m$ it is not possible to compute all $2^m$ terms. There is a way to avoid this exponential explosion if we can assume that a small number of terms dominate the sums. If, for instance, we assume that the probability that more than $k$ hidden units are active ($+1$) at the same time is negligibly small we can get a good approximation by computing only $O(m^k)$ terms. Alternately, if we are not sure which states of the hidden units have non-negligible probability, we can dynamically search, as part of the learning process, for the significant terms in the sum. This way we get an algorithm that is always accurate, and is efficient when the number of significant terms is small. In the extreme case where we assume that only one hidden unit is active at a time (i.e. $k = 1$), the harmonium model essentially reduces to the standard mixture model as discussed is section 3. For larger $k$, this type of assumption provides a middle ground between the generality of the harmonium model and the simplicity of the mixture model.

## 5   Projection Pursuit methods

A statistical method that has a close relationship with the harmonium model is the Projection Pursuit (PP) technique [10,6;5]. The use of projection pursuit in the context of neural networks has been studied by several researchers (e.g. [11]). Most of the work is in *exploratory* projection pursuit and projection pursuit *regression*. In this paper we are interested in projection pursuit *density estimation*. Here PP avoids the exponential blowup of the standard gradient ascent technique, and also has that advantage that the number $m$ of hidden units is estimated from the sample as well, rather than being specified in advance.

Projection pursuit density estimation [6] is based on several types of analysis, using the central limit theorem, that lead to the following general conclusion. *If $\vec{x} \in R^n$ is a random vector for which the different coordinates are independent, and $\vec{\omega} \in R^n$ is a vector from the n dimensional unit sphere, then the distribution of the projection $\vec{\omega} \cdot \vec{x}$ is close to gaussian for most $\vec{\omega}$.* Thus searching for those directions $\vec{\omega}$ for which the projection of a sample is most non-gaussian is a way for detecting dependencies between the coordinates in high dimensional distributions. Several "projection-indices" have been studied in the literature for measuring the "non-gaussianity" of projection, each enhancing different properties of the projected distribution. In order to find more than one projection direction, several methods of "structure elimination" have been devised. These methods transform the sample in such a way that the the direction in which non-gaussianity has been detected appears to be gaussian, thus enabling the algorithm to detect non-gaussian projections that would otherwise be obscured. The search for a description of the distribution of a sample in terms of its projections can be formalized in the context of maximal likelihood density estimation [6]. In order to create a formal relation between the harmonium model and projection pursuit, we define a variant of the model that defines a density over $R^n$ instead of a distribution over $\{\pm1\}^n$. Based on this form we devise a projection index and a structure removal method that are the basis of the following learning algorithm (described fully in [4])

- **Initialization**
  Set $S_0$ to be the input sample.
  Set $p_0$ to be the initial distribution (Gaussian).

- **Iteration**
  Repeat the following steps for $i = 1, 2 \ldots$ until no single-variable harmonium model has a significantly higher likelihood than the Gaussian distribution with respect to $S_i$.

  1. Perform an estimate-maximize (EM) [2] search on the log-likelihood of a single hidden variable model on the sample $S_{i-1}$. Denote by $\theta_i$ and $\vec{\omega}^{(i)}$ the parameters found by the search, and create a new hidden unit with associated binary r.v. $h_i$ with these weights and bias.

  2. Transform $S_{i-1}$ into $S_i$ using the following structure removal procedure.
     For each example $\vec{z} \in S_{i-1}$ compute the probability that the hidden variable $h_i$ found in the last step is 1 on this input:

     $$P(h_i = 1) = \left(1 + e^{-(\theta_i + \vec{\omega}^{(i)} \cdot \vec{z})}\right)^{-1}$$

     Flip a coin that has probability of "head" equal to $P(h_i = 1)$. If the coin turns out "head" then add $\vec{z} - \vec{\omega}^{(i)}$ to $S_i$ else add $\vec{z}$ to $S_i$.

  3. Set $p_i(\vec{z})$ to be $p_{i-1}(\vec{z}) Z_i^{-1} \left(1 + e^{\theta_i + \vec{\omega}^{(i)} \cdot \vec{z}}\right)$.

## 6   Experimental work

We have carried out several experiments to test the performance of unsupervised learning using the harmonium model. These are not, at this stage, extensive experimental comparisons, but they do provide initial insights into the issues regarding our learning algorithms and the use of the harmonium model for learning real world tasks.

The first set of experiments studies two methods for learning the harmonium model. The first is the gradient ascent method, and the second is the projection pursuit method. The experiments in this set were performed on synthetically generated data. The input consisted of binary vectors of 64 bits that represent $8 \times 8$ binary images. The images are synthesized using a harmonium model with 10 hidden units whose weights were set as in Figure (2,a). The ultimate goal of the learning algorithms was to retrieve the model that generated the data. To measure the quality of the models generated by the algorithms we use three different measures. The likelihood of the model, [6] the fraction of correct predictions the model makes when used to predict the value of a single input bit given all the other bits, and the performance of the model when used to reconstruct the input from the most probable state of the hidden units. [7] All experiments use a test set and a train set, each containing 1000 examples. The gradient ascent method used a standard momentum term, and typically needed about 1000 epochs to stabilize. In the projection pursuit algorithm, 4 iterations of EM per hidden unit proved sufficient to find a stable solution. The results are summarized in the following table and in Figure (2).

|  | likelihood | | single bit prediction | | input reconstruction | |
|---|---|---|---|---|---|---|
|  | train | test | train | test | train | test |
| gradient ascent for 1000 epochs | 0.399 | 0.425 | 0.098 | 0.100 | 0.311 | 0.338 |
| projection pursuit | 0.799 | 0.802 | 0.119 | 0.114 | 0.475 | 0.480 |
| Projection pursuit followed by gradient ascent for 100 epochs | 0.411 | 0.430 | 0.091 | 0.089 | 0.315 | 0.334 |
| Projection pursuit followed by gradient ascent for 1000 epochs | 0.377 | 0.405 | 0.071 | 0.082 | 0.261 | 0.287 |
| true model | 0.372 | 0.404 | 0.062 | 0.071 | 0.252 | 0.283 |

Looking at the table and Figure (2), and taking into account execution times, it appears that gradient ascent is slow but eventually finds much of the underlying structure in the distribution, although several of the hidden units (see units 1,2,6,7, counting from the left, in Figure (2,a)) have no obvious relation to the true model. In contrast, PP is fast and finds all of the features of the true model albeit sometimes

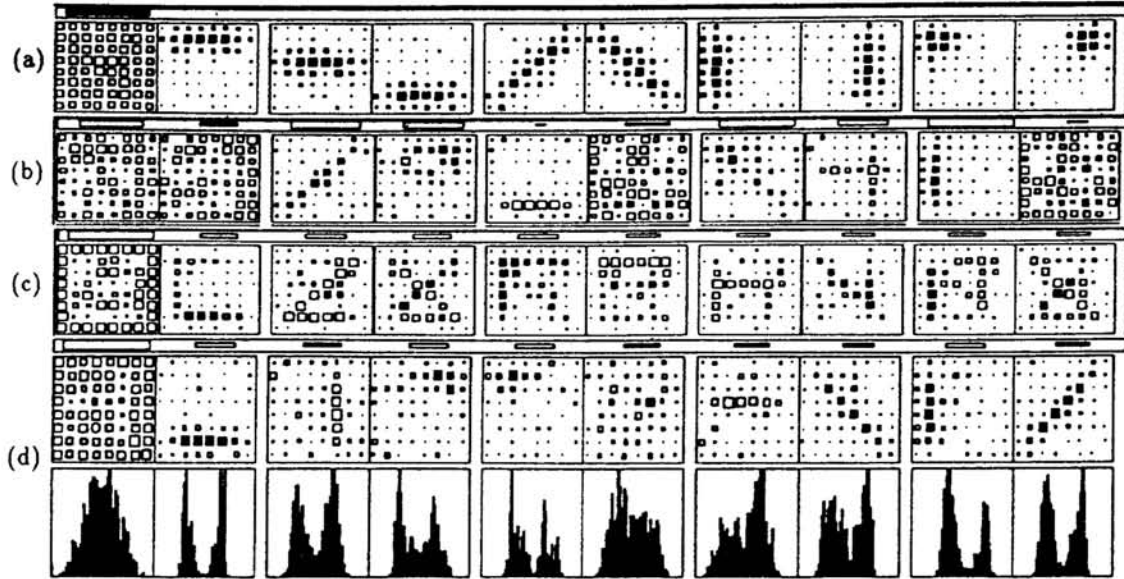

Figure 2: The weight vectors of the models in the synthetic data experiments. Each matrix represents the 64 weights of one hidden unit. The square above the matrix represents the units bias. positive weights are displayed as full squares and negative weights as empty squares, the area of the square is proportional to the absolute value of the weight. (a) The weights in the model found by gradient ascent alone. (b) The weights in the model found by projection pursuit alone. (c) The weights in the model used for generating the data. (d) The weights in the model found by projection pursuit followed by gradient ascent. For this last model we also show the histograms of the projection of the examples on the directions defined by those weight vectors; the bimodality expected from projection pursuit analysis is evident.

in combinations, However, the error measurements show that something is still missing from the models found by our implementation of PP. Following PP by a gradient ascent phase seems to give the best of both algorithms, finding a good approximation after only 140 epochs (40 PP + 100 gradient) and recovering the true model almost exactly after 1040 epochs.

In the second set of experiments we compare the performance of the harmonium model to that of the mixture model. The comparison uses real world data extracted from the NIST handwritten data base [8], Examples are 16 × 16 binary images (see Figure (3)). We use 60 hidden units to model the distribution in both of the models. Because of the large number of hidden units we cannot use gradient ascent learning and instead use projection pursuit. For the same reason it was not possible to compute the likelihood of the harmonium model and only the other two measures of error were used. Each test was run several times to get accuracy bounds on the measurements. The results are summarized in the following table

|  | single bit prediction | | input reconstruction | |
|---|---|---|---|---|
|  | train | test | train | test |
| Mixture model | $0.185 \pm 0.005$ | $0.258 \pm 0.005$ | $0.518 \pm 0.002$ | $0.715 \pm 0.002$ |
| Harmonium model | $0.20 \pm 0.01$ | $0.21 \pm 0.01$ | $0.63 \pm 0.05$ | $0.66 \pm 0.03$ |

In Figure (4) we show some typical weight vectors found for the mixture model and for the harmonium model, it is clear that while the mixture model finds weights that are some kind of average prototypes of complete digits, the harmonium model finds weights that correspond to local features such as lines and contrasts. There is a small but definite improvement in the errors of the harmonium model with respect to the errors of the mixture model. As the experiments on synthetic data have shown that PP does not reach

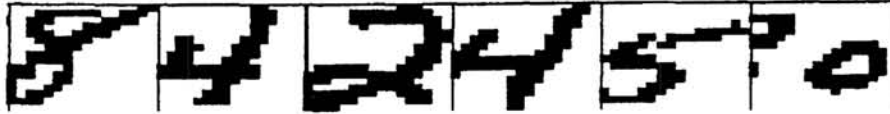

Figure 3: A few examples from the handwritten digits sample.

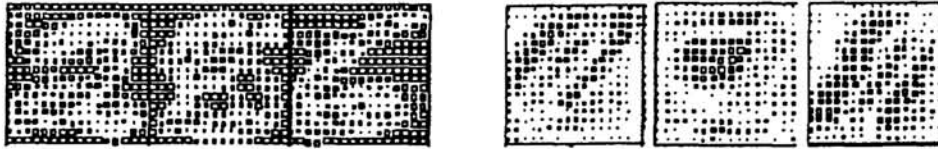

Figure 4: Typical weight vectors found by the mixture model (left) and the harmonium model (right)

optimal solutions by itself we expect the advantage of the harmonium model over the mixture model will increase further by using improved learning methods. Of course, the harmonium model is a very general distribution model and is not specifically tuned to the domain of handwritten digit images, thus it cannot be compared to models specifically developed to capture structures in this domain. However, the experimental results supports our claim that the harmonium model is a simple and tractable mathematical model for describing distributions in which several correlation patterns combine to generate each individual example.

## Footnotes

*yoav@cis.ucsc.edu

[1] One possible solution to this is the mean-field approximation [15], discussed further in section 2 below.

[2] In [4] we show that any distribution over $\{\pm1\}^n$ can be approximated to within any desired accuracy by a harmonium model using $2^n$ hidden units.

[3] See also other work relating Bayes nets and Boltzmann machines [12,7].

[4] Noisy-OR gates have been introduced in the framework of Bayes Networks to allow for such combinations. However, using this in networks with hidden units has not been studied, to the best of our knowledge.

[5] In [16][Ch.6], binary connection weights are used. Here we use real-valued weights.

[6] We present the negation of the log-likelihood, scaled so that the uniform distribution will have likelihood 1.0

[7] More precisely, for each input unit $i$ we compute the probability $p_i$ that it has value $+1$. Then for example $(x_1, \ldots, x_n)$, we measure $-\sum_{i=1}^{n} \log_2(1/2 + x_i(p_i - 1/2))$.

[8]NIST Special Database 1, HWDB Rel1-1.1, May 1990.

## References

[1] D. H. Ackley, G. E. Hinton, and T. J. Sejnowski. A learning algorithm for Boltzmann machines. *Cognitive Science*, 9:147–169, 1985.

[2] A. Dempster, N. Laird, and D. Rubin. Maximum likelihood from incomplete data via the EM algorithm. *J. Roy. Statist. Soc. B*, 39:1–38, 1977.

[3] B. Everitt and D. Hand. *Finite mixture distributions*. Chapman and Hall, 1981.

[4] Y. Freund and D. Haussler. Unsupervised learning of distributions on binary vectors using two layer networks. Technical Report UCSC-CRL-91-20, Univ. of Calif. Computer Research Lab, Santa Cruz, CA, 1992 (To appear).

[5] J. H. Friedman. Exploratory projection pursuit. *J. Amer. Stat.Assoc.*, 82(397):599–608, Mar. 1987.

[6] J. H. Friedman, W.Stuetzle, and A. Schroeder. Projection pursuit density estimation. *J. Amer. Stat.Assoc.*, 79:599–608, 1984.

[7] H. Gefner and J. Pearl. On the probabilistic semantics of connectionist networks. Technical Report CSD-870033, UCLA Computer Science Department, July 1987.

[8] S. Geman and D. Geman. Stochastic relaxations, Gibbs distributions and the Bayesian restoration of images. *IEEE Trans. on Pattern Analysis and Machine Intelligence*, 6:721–742, 1984.

[9] J. Hopfield. Neural networks and physical systems with emergent collective computational abilities. *Proc. Natl. Acad Sci. USA*, 79:2554–2558, Apr. 1982.

[10] P. Huber. Projection pursuit (with discussion). *Ann. Stat.*, 13:435–525, 1985.

[11] N. Intrator. Feature extraction using an unsupervised neural network. In D. Touretzky, J. Ellman, T. Sejnowski, and G. Hinton, editors, *Proceedings of the 1990 Connectionist Models Summer School*, pages 310–318. Morgan Kaufmann, San Mateo, CA., 1990.

[12] R. M. Neal. Learning stochastic feedforward networks. Technical report, Department of Computer Science, University of Toronto, Nov. 1990.

[13] S. Nowlan. Maximum likelihood competitive learning. In D. Touretsky, editor, *Advances in Neural Information Processing Systems*, volume 2, pages 574–582. Morgan Kaufmann, 1990.

[14] J. Pearl. *Probabilistic Reasoning in Intelligent Systems*. Morgan Kaufmann, 1988.

[15] C. Peterson and J. R. Anderson. A mean field theory learning algorithm for neural networks. *Complex Systems*, 1:995–1019, 1987.

[16] D. E. Rumelhart and J. L. McClelland. *Parallel Distributed Processing: Explorations in the Microstructure of Cognition. Volume 1: Foundations*. MIT Press, Cambridge, Mass., 1986.
